# A Geometric take on Metric Learning

**Søren Hauberg**
MPI for Intelligent Systems
Tübingen, Germany
soren.hauberg@tue.mpg.de

**Oren Freifeld**
Brown University
Providence, US
freifeld@dam.brown.edu

**Michael J. Black**
MPI for Intelligent Systems
Tübingen, Germany
black@tue.mpg.de

## Abstract

Multi-metric learning techniques learn local metric tensors in different parts of a feature space. With such an approach, even simple classifiers can be competitive with the state-of-the-art because the distance measure locally adapts to the structure of the data. The learned distance measure is, however, non-metric, which has prevented multi-metric learning from generalizing to tasks such as dimensionality reduction and regression in a principled way. We prove that, with appropriate changes, multi-metric learning corresponds to learning the structure of a Riemannian manifold. We then show that this structure gives us a principled way to perform dimensionality reduction and regression according to the learned metrics. Algorithmically, we provide the first practical algorithm for computing geodesics according to the learned metrics, as well as algorithms for computing exponential and logarithmic maps on the Riemannian manifold. Together, these tools let many Euclidean algorithms take advantage of multi-metric learning. We illustrate the approach on regression and dimensionality reduction tasks that involve predicting measurements of the human body from shape data.

## 1 Learning and Computing Distances

Statistics relies on measuring distances. When the Euclidean metric is insufficient, as is the case in many real problems, standard methods break down. This is a key motivation behind *metric learning*, which strives to learn good distance measures from data. In the most simple scenarios a single *metric tensor* is learned, but in recent years, several methods have proposed learning *multiple* metric tensors, such that different distance measures are applied in different parts of the feature space. This has proven to be a very powerful approach for classification tasks [1, 2], but the approach has not generalized to other tasks. Here we consider the generalization of *Principal Component Analysis (PCA)* and *linear regression*; see Fig. 1 for an illustration of our approach. The main problem with generalizing multi-metric learning is that it is based on assumptions that make the feature space both non-smooth and non-metric. Specifically, it is often assumed that straight lines form geodesic curves and that the metric tensor stays constant along these lines. These assumptions are made because it is believed that computing the actual geodesics is intractable, requiring a discretization of the entire feature space [3]. We solve these problems by smoothing the transitions between different metric tensors, which ensures a metric space where geodesics can be computed.

In this paper, we consider the scenario where the metric tensor at a given point in feature space is defined as the weighted average of a set of learned metric tensors. In this model, we prove that the feature space becomes a chart for a *Riemannian manifold*. This ensures a metric feature space, i.e.

$$\mathrm{dist}(\mathbf{x}, \mathbf{y}) = 0 \iff \mathbf{x} = \mathbf{y} \ ,$$
$$\mathrm{dist}(\mathbf{x}, \mathbf{y}) = \mathrm{dist}(\mathbf{y}, \mathbf{x}) \qquad \text{(symmetry)}, \tag{1}$$
$$\mathrm{dist}(\mathbf{x}, \mathbf{z}) \leq \mathrm{dist}(\mathbf{x}, \mathbf{y}) + \mathrm{dist}(\mathbf{y}, \mathbf{z}) \qquad \text{(triangle inequality)}.$$

To compute statistics according to the learned metric, we need to be able to compute distances, which implies that we need to compute geodesics. Based on the observation that geodesics are

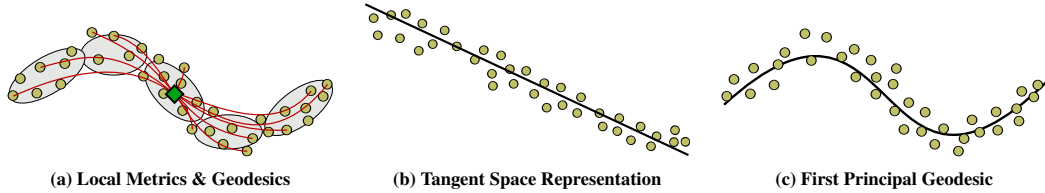

<div align="center">

**(a) Local Metrics & Geodesics**    **(b) Tangent Space Representation**    **(c) First Principal Geodesic**

</div>

Figure 1: Illustration of Principal Geodesic Analysis. (a) Geodesics are computed between the mean and each data point. (b) Data is mapped to the Euclidean tangent space and the first principal component is computed. (c) The principal component is mapped back to the feature space.

smooth curves in Riemannian spaces, we derive an algorithm for computing geodesics that only requires a discretization of the geodesic rather than the entire feature space. Furthermore, we show how to compute the *exponential* and *logarithmic maps* of the manifold. With this we can map any point back and forth between a Euclidean tangent space and the manifold. This gives us a general strategy for incorporating the learned metric tensors in many Euclidean algorithms: map the data to the tangent of the manifold, perform the Euclidean analysis and map the results back to the manifold.

Before deriving the algorithms (Sec. 3) we set the scene by an analysis of the shortcomings of current state-of-the-art methods (Sec. 2), which motivate our final model. The model is general and can be used for many problems. Here we illustrate it with several challenging problems in 3D body shape modeling and analysis (Sec. 4). All proofs can be found in the supplementary material along with algorithmic details and further experimental results.

## 2 Background and Related Work

Single-metric learning learns a metric tensor, $\mathbf{M}$, such that distances are measured as

$$\text{dist}^2(\mathbf{x}_i, \mathbf{x}_j) = \|\mathbf{x}_i - \mathbf{x}_j\|_{\mathbf{M}}^2 \equiv (\mathbf{x}_i - \mathbf{x}_j)^T \mathbf{M}(\mathbf{x}_i - \mathbf{x}_j) \ , \tag{2}$$

where $\mathbf{M}$ is a symmetric and positive definite $D \times D$ matrix. Classic approaches for finding such a metric tensor include PCA, where the metric is given by the inverse covariance matrix of the training data; and *linear discriminant analysis (LDA)*, where the metric tensor is $\mathbf{M} = \mathbf{S}_W^{-1}\mathbf{S}_B\mathbf{S}_W^{-1}$, with $\mathbf{S}_w$ and $\mathbf{S}_B$ being the *within class scatter* and the *between class scatter* respectively [9].

A more recent approach tries to learn a metric tensor from triplets of data points $(\mathbf{x}_i, \mathbf{x}_j, \mathbf{x}_k)$, where the metric should obey the constraint that $\text{dist}(\mathbf{x}_i, \mathbf{x}_j) < \text{dist}(\mathbf{x}_i, \mathbf{x}_k)$. Here the constraints are often chosen such that $\mathbf{x}_i$ and $\mathbf{x}_j$ belong to the same class, while $\mathbf{x}_i$ and $\mathbf{x}_k$ do not. Various relaxed versions of this idea have been suggested such that the metric can be learned by solving a semi-definite or a quadratic program [1, 2, 4–8]. Among the most popular approaches is the *Large Margin Nearest Neighbor (LMNN)* classifier [5], which finds a linear transformation that satisfies local distance constraints, making the approach suitable for multi-modal classes.

For many problems, a single *global* metric tensor is not enough, which motivates learning several *local* metric tensors. The classic work by Hastie and Tibshirani [9] advocates locally learning metric tensors according to LDA and using these as part of a $k$NN classifier. In a somewhat similar fashion, Weinberger and Saul [5] cluster the training data and learn a separate metric tensor for each cluster using LMNN. A more extreme point of view was taken by Frome et al. [1, 2], who learn a diagonal metric tensor for every point in the training set, such that distance rankings are preserved. Similarly, Malisiewicz and Efros [6] find a diagonal metric tensor for each training point such that the distance to a subset of the training data from the same class is kept small.

Once a set of metric tensors $\{\mathbf{M}_1, \ldots, \mathbf{M}_R\}$ has been learned, the distance $\text{dist}(\mathbf{a}, \mathbf{b})$ is measured according to (2) where "the nearest" metric tensor is used, i.e.

$$\mathbf{M}(\mathbf{x}) = \sum_{r=1}^{R} \frac{\tilde{w}_r(\mathbf{x})}{\sum_j \tilde{w}_j(\mathbf{x})}\mathbf{M}_r \ , \quad \text{where} \ \tilde{w}_r(\mathbf{x}) = \begin{cases} 1 & \|\mathbf{x} - \mathbf{x}_r\|_{\mathbf{M}_r}^2 \leq \|\mathbf{x} - \mathbf{x}_j\|_{\mathbf{M}_j}^2, \forall j \\ 0 & \text{otherwise} \end{cases} \ , \tag{3}$$

where $\mathbf{x}$ is either $\mathbf{a}$ or $\mathbf{b}$ depending on the algorithm. Note that this gives a non-metric distance function as it is not symmetric. To derive this equation, it is necessary to assume that 1) geodesics

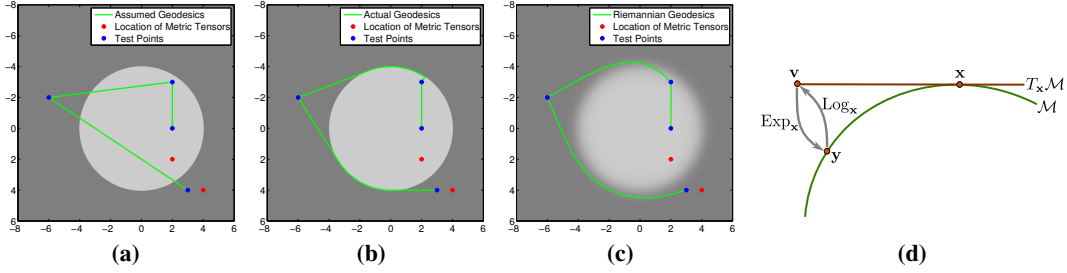

Figure 2: (a)–(b) An illustrative example where straight lines do not form geodesics and where the metric tensor does not stay constant along lines; see text for details. The background color is proportional to the trace of the metric tensor, such that light grey corresponds to regions where paths are short ($\mathbf{M}_1$), and dark grey corresponds to regions they are long ($\mathbf{M}_2$). (c) The suggested geometric model along with the geodesics. Again, background colour is proportional to the trace of the metric tensor; the colour scale is the same is used in (a) and (b). (d) An illustration of the exponential and logarithmic maps.

form straight lines, and 2) the metric tensor stays constant along these lines [3]. Both assumptions are problematic, which we illustrate with a simple example in Fig. 2a–c.

Assume we are given two metric tensors $\mathbf{M}_1 = 2\mathbf{I}$ and $\mathbf{M}_2 = \mathbf{I}$ positioned at $\mathbf{x}_1 = (2, 2)^T$ and $\mathbf{x}_2 = (4, 4)^T$ respectively. This gives rise to two regions in feature space in which $\mathbf{x}_1$ is nearest in the first and $\mathbf{x}_2$ is nearest in the second, according to (3). This is illustrated in Fig. 2a. In the same figure, we also show the assumed straight-line geodesics between selected points in space. As can be seen, two of the lines goes through both regions, such that the assumption of constant metric tensors along the line is violated. Hence, it would seem natural to measure the length of the line, by adding the length of the line segments which pass through the different regions of feature space. This was suggested by Ramanan and Baker [3] who also proposed a polynomial time algorithm for measuring these line lengths. This gives a symmetric distance function.

Properly computing line lengths according to the local metrics is, however, not enough to ensure that the distance function is metric. As can be seen in Fig. 2a the straight line does *not* form a geodesic as a shorter path can be found by circumventing the region with the "expensive" metric tensor $\mathbf{M}_1$ as illustrated in Fig. 2b. This issue makes it trivial to construct cases where the triangle inequality is violated, which again makes the line length measure non-metric.

In summary, if we want a metric feature space, we can neither assume that geodesics are straight lines nor that the metric tensor stays constant along such lines. In practice, good results have been reported using (3) [1,3,5], so it seems obvious to ask: *is metricity required?* For kNN classifiers this does not appear to be the case, with many successes based on dissimilarities rather than distances [10]. We, however, want to generalize *PCA* and *linear regression*, which both seek to minimize the reconstruction error of points projected onto a subspace. As the notion of *projection* is hard to define sensibly in non-metric spaces, we consider metricity essential.

In order to build a model with a metric feature space, we change the weights in (3) to be smooth functions. This impose a well-behaved geometric structure on the feature space, which we take advantage of in order to perform statistical analysis according to the learned metrics. However, first we review the basics of Riemannian geometry as this provides the theoretical foundation of our work.

## 2.1 Geodesics and Riemannian Geometry

We start by defining Riemannian manifolds, which intuitively are smoothly curved spaces equipped with an inner product. Formally, they are smooth manifolds endowed with a *Riemannian metric* [11]:

**Definition** A *Riemannian metric* $\mathbf{M}$ on a manifold $\mathcal{M}$ is a smoothly varying inner product $< \mathbf{a}, \mathbf{b} >_{\mathbf{x}} = \mathbf{a}^T \mathbf{M}(\mathbf{x})\mathbf{b}$ in the tangent space $T_{\mathbf{x}}\mathcal{M}$ of each point $\mathbf{x} \in \mathcal{M}$ .

Often Riemannian manifolds are represented by a *chart*; i.e. a parameter space for the curved surface. An example chart is the spherical coordinate system often used to represent spheres. While such charts are often flat spaces, the curvature of the manifold arises from the smooth changes in the metric.

On a Riemannian manifold $\mathcal{M}$, the length of a smooth curve $\mathbf{c} : [0,1] \to \mathcal{M}$ is defined as the integral of the norm of the tangent vector (interpreted as *speed*) along the curve:

$$\text{Length}(\mathbf{c}) = \int_0^1 \|\mathbf{c}'(\lambda)\|_{\mathbf{M}(\mathbf{c}(\lambda))} \mathrm{d}\lambda = \int_0^1 \sqrt{\mathbf{c}'(\lambda)^T \mathbf{M}(\mathbf{c}(\lambda)) \mathbf{c}'(\lambda)} \mathrm{d}\lambda \ , \qquad (4)$$

where $\mathbf{c}'$ denotes the derivative of $\mathbf{c}$ and $\mathbf{M}(\mathbf{c}(\lambda))$ is the metric tensor at $\mathbf{c}(\lambda)$. A *geodesic* curve is then a length-minimizing curve connecting two given points $\mathbf{x}$ and $\mathbf{y}$, i.e.

$$\mathbf{c}_{\text{geo}} = \underset{\mathbf{c}}{\arg\min} \left( \text{Length}(\mathbf{c}) \right) \quad \text{with} \quad \mathbf{c}(0) = \mathbf{x} \quad \text{and} \quad \mathbf{c}(1) = \mathbf{y} \ . \qquad (5)$$

The distance between $\mathbf{x}$ and $\mathbf{y}$ is defined as the length of the geodesic.

Given a tangent vector $\mathbf{v} \in T_\mathbf{x}\mathcal{M}$, there exists a unique geodesic $\mathbf{c}_\mathbf{v}(t)$ with initial velocity $\mathbf{v}$ at $\mathbf{x}$. The Riemannian exponential map, $\text{Exp}_\mathbf{x}$, maps $\mathbf{v}$ to a point on the manifold along the geodesic $\mathbf{c}_\mathbf{v}$ at $t = 1$. This mapping preserves distances such that $\text{dist}(\mathbf{c}_\mathbf{v}(0), \mathbf{c}_\mathbf{v}(1)) = \|\mathbf{v}\|$. The inverse of the exponential map is the Riemannian *logarithmic map* denoted $\text{Log}_\mathbf{x}$. Informally, the exponential and logarithmic maps move points back and forth between the manifold and the tangent space while preserving distances (see Fig. 2d for an illustration). This provides a general strategy for generalizing many Euclidean techniques to Riemannian domains: data points are mapped to the tangent space, where ordinary Euclidean techniques are applied and the results are mapped back to the manifold.

## 3 A Metric Feature Space

With the preliminaries settled we define the new model. Let $\mathcal{C} = \mathbb{R}^D$ denote the feature space. We endow $\mathcal{C}$ with a metric tensor in every point $\mathbf{x}$, which we define akin to (3),

$$\mathbf{M}(\mathbf{x}) = \sum_{r=1}^{R} w_r(\mathbf{x}) \mathbf{M}_r \ , \quad \text{where} \quad w_r(\mathbf{x}) = \frac{\tilde{w}_r(\mathbf{x})}{\sum_{j=1}^{R} \tilde{w}_j(\mathbf{x})} \ , \qquad (6)$$

with $\tilde{w}_r > 0$. The only difference from (3) is that we shall not restrict ourselves to binary weight functions $\tilde{w}_r$. We assume the metric tensors $\mathbf{M}_r$ have already been learned; Sec. 4 contain examples where they have been learned using LMNN [5] and LDA [9].

From the definition of a Riemannian metric, we trivially have the following result:

**Lemma 1** *The space $\mathcal{C} = \mathbb{R}^D$ endowed with the metric tensor from (6) is a chart of a Riemannian manifold, iff the weights $w_r(\mathbf{x})$ change smoothly with $\mathbf{x}$.*

Hence, by only considering smooth weight functions $\tilde{w}_r$ we get a well-studied geometric structure on the feature space, which ensures us that it is metric. To illustrate the implications we return to the example in Fig. 2. We change the weight functions from binary to squared exponentials, which gives the feature space shown in Fig. 2c. As can be seen, the metric tensor now changes smoothly, which also makes the geodesics smooth curves (a property we will use when computing the geodesics).

It is worth noting that Ramanan and Baker [3] also consider the idea of smoothly averaging the metric tensor. They, however, only evaluate the metric tensor at the test point of their classifier and then assume straight line geodesics with a constant metric tensor. Such assumptions violate the premise of a smoothly changing metric tensor and, again, the distance measure becomes non-metric.

Lemma 1 shows that metric learning can be viewed as *manifold learning*. The main difference between our approach and techniques such as *Isomap* [12] is that, while Isomap learns an embedding of the data points, we learn the actual manifold structure. This gives us the benefit that we can compute geodesics as well as the exponential and logarithmic maps. These provide us with mappings back and forth between the manifold and Euclidean representation of the data, which preserve distances as well as possible. The availability of such mappings is in stark contrast to e.g. Isomap.

In the next section we will derive a system of ordinary differential equations (ODE's) that geodesics in $\mathcal{C}$ have to satisfy, which provides us with algorithms for computing geodesics as well as exponential and logarithmic maps. With these we can generalize many Euclidean techniques.

### 3.1 Computing Geodesics, Maps and Statistics

At minima of (4) we know that the Euler-Lagrange equation must hold [11], i.e.

$$\frac{\partial L}{\partial \mathbf{c}} = \frac{\mathrm{d}}{\mathrm{d}\lambda}\frac{\partial L}{\partial \mathbf{c}'} \ , \ \text{ where } \ L(\lambda, \mathbf{c}, \mathbf{c}') = \mathbf{c}'(\lambda)^T \mathbf{M}(\mathbf{c}(\lambda))\mathbf{c}'(\lambda) \ . \tag{7}$$

As we have an explicit expression for the metric tensor we can compute (7) in closed form:

**Theorem 2** *Geodesic curves in $\mathcal{C}$ satisfy the following system of $2^{\mathrm{nd}}$ order ODE's*

$$\mathbf{M}(\mathbf{c}(\lambda))\mathbf{c}''(\lambda) = -\frac{1}{2}\left[\frac{\partial \mathsf{vec}\left[\mathbf{M}(\mathbf{c}(\lambda))\right]}{\partial \mathbf{c}(\lambda)}\right]^T (\mathbf{c}'(\lambda) \otimes \mathbf{c}'(\lambda)) \ , \tag{8}$$

*where $\otimes$ denotes the Kronecker product and $\mathsf{vec}\left[\cdot\right]$ stacks the columns of a matrix into a vector [13].*

**Proof** See supplementary material. □

This result holds for any smooth weight functions $\tilde{w}_r$. We, however, still need to compute $\frac{\partial \mathsf{vec}[\mathbf{M}]}{\partial \mathbf{c}}$, which depends on the specific choice of $\tilde{w}_r$. Any smooth weighting scheme is applicable, but we restrict ourselves to the obvious smooth generalization of (3) and use squared exponentials. From this assumption, we get the following result

**Theorem 3** *For $\tilde{w}_r(\mathbf{x}) = \exp\left(-\frac{\varrho}{2}\|\mathbf{x} - \mathbf{x}_r\|_{\mathbf{M}_r}^2\right)$ the derivative of the metric tensor from (6) is*

$$\frac{\partial \mathsf{vec}\left[\mathbf{M}(\mathbf{c})\right]}{\partial \mathbf{c}} = \frac{\rho}{\left(\sum_{j=1}^{R}\tilde{w}_j\right)^2}\sum_{r=1}^{R}\tilde{w}_r\mathsf{vec}\left[\mathbf{M}_r\right]\sum_{j=1}^{R}\tilde{w}_j\left((\mathbf{c}-\mathbf{x}_j)^T\mathbf{M}_j - (\mathbf{c}-\mathbf{x}_r)^T\mathbf{M}_r\right) \ . \tag{9}$$

**Proof** See supplementary material. □

**Computing Geodesics.** Any geodesic curve must be a solution to (8). Hence, to compute a geodesic between $\mathbf{x}$ and $\mathbf{y}$, we can solve (8) subject to the constraints

$$\mathbf{c}(0) = \mathbf{x} \ \text{ and } \ \mathbf{c}(1) = \mathbf{y} \ . \tag{10}$$

This is a *boundary value problem*, which has a smooth solution. This allows us to solve the problem numerically using a standard three-stage Lobatto IIIa formula, which provides a fourth-order accurate $C^1$–continuous solution [14].

Ramanan and Baker [3] discuss the possibility of computing geodesics, but arrive at the conclusion that this is intractable based on the assumption that it requires discretizing the entire feature space. *Our solution avoids discretizing the feature space by discretizing the geodesic curve instead. As this is always one-dimensional the approach remains tractable in high-dimensional feature spaces.*

**Computing Logarithmic Maps.** Once a geodesic $\mathbf{c}$ is found, it follows from the definition of the logarithmic map, $\mathrm{Log}_{\mathbf{x}}(\mathbf{y})$, that it can be computed as

$$\mathbf{v} = \mathrm{Log}_{\mathbf{x}}(\mathbf{y}) = \frac{\mathbf{c}'(0)}{\|\mathbf{c}'(0)\|}\mathrm{Length}(\mathbf{c}) \ . \tag{11}$$

In practice, we solve (8) by rewriting it as a system of first order ODE's, such that we compute both $\mathbf{c}$ and $\mathbf{c}'$ simultaneously (see supplementary material for details).

**Computing Exponential Maps.** Given a starting point $\mathbf{x}$ on the manifold and a vector $\mathbf{v}$ in the tangent space, the exponential map, $\mathrm{Exp}_{\mathbf{x}}(\mathbf{v})$, finds the unique geodesic starting at $\mathbf{x}$ with initial velocity $\mathbf{v}$. As the geodesic must fulfill (8), we can compute the exponential map by solving this system of ODE's with the initial conditions

$$\mathbf{c}(0) = \mathbf{x} \ \text{ and } \ \mathbf{c}'(0) = \mathbf{v} \ . \tag{12}$$

This *initial value problem* has a unique solution, which we find numerically using a standard Runge-Kutta scheme [15].

### 3.1.1 Generalizing PCA and Regression

At this stage, we know that the feature space is Riemannian and we know how to compute geodesics and exponential and logarithmic maps. We now seek to generalize PCA and linear regression, which becomes straightforward since solutions are available in Riemannian spaces [16, 17]. These generalizations can be summarized as mapping the data to the tangent space at the mean, performing standard Euclidean analysis in the tangent and mapping the results back.

The first step is to compute the mean value on the manifold, which is defined as the point that minimizes the sum-of-squares distances to the data points. Pennec [18] provides an efficient gradient descent approach for computing this point, which we also summarize in the supplementary material.

The empirical covariance of a set of points is defined as the ordinary Euclidean covariance in the tangent space at the mean value [18]. With this in mind, it is not surprising that the principal components of a dataset have been generalized as the geodesics starting at the mean with initial velocity corresponding to the eigenvectors of the covariance [16],

$$\gamma_{\mathbf{v}_d}(t) = \mathrm{Exp}_\mu(t\mathbf{v}_d) \ , \tag{13}$$

where $\mathbf{v}_d$ denotes the $d^{\mathrm{th}}$ eigenvector of the covariance. This approach is called *Principal Geodesic Analysis (PGA)*, and the geodesic curve $\gamma_{\mathbf{v}_d}$ is called the *principal geodesic*. An illustration of the approach can be seen in Fig. 1 and more algorithmic details are in the supplementary material.

Linear regression has been generalized in a similar way [17] by performing regression in the tangent of the mean and mapping the resulting line back to the manifold using the exponential map.

The idea of working in the tangent space is both efficient and convenient, but comes with an element of approximation as the logarithmic map is only guaranteed to preserve distances to the origin of the tangent and not between all pairs of data points. Practical experience, however, indicates that this is a good tradeoff; see [19] for a more in-depth discussion of when the approximation is suitable.

## 4 Experiments

To illustrate the framework[1] we consider an example in human body analysis, and then we analyze the scalability of the approach. But first, to build intuition, Fig. 3a show synthetically generated data samples from two classes. We sample random points $\mathbf{x}_r$ and learn a local LDA metric [9] by considering all data points within a radius; this locally pushes the two classes apart. We combine the local metrics using (6) and Fig. 3b show the data in the tangent space of the resulting manifold. As can be seen the two classes are now globally further apart, which shows the effect of local metrics.

### 4.1 Human Body Shape

We consider a regression example concerning human body shape analysis. We study 986 female body laser scans from the CAESAR [20] data set; each shape is represented using the leading 35 principal components of the data learned using a SCAPE-like model [21, 22]. Each shape is associated with anthropometric measurements such as *body height*, *shoe size*, etc. We show results for *shoulder to wrist distance* and *shoulder breadth*, but results for more measurements are in the supplementary material. To predict the measurements from shape coefficients, we learn local metrics and perform linear regression according to these. As a further experiment, we use PGA to reduce the dimensionality of the shape coefficients according to the local metrics, and measure the quality of the reduction by performing linear regression to predict the measurements. As a baseline we use the corresponding Euclidean techniques.

To learn the local metric we do the following. First we whiten the data such that the variance captured by PGA will only be due to the change of metric; this allows easy visualization of the impact of the learned metrics. We then cluster the body shapes into equal-sized clusters according to the measurement and learn a LMNN metric for each cluster [5], which we associate with the mean of each class. These push the clusters apart, which introduces variance along the directions where the measurement changes. From this we construct a Riemannian manifold according to (6),

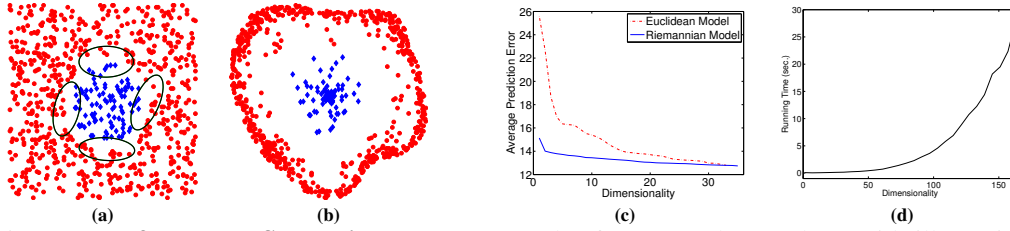

Figure 3: **Left panels: Synthetic data.** (a) Samples from two classes along with illustratively sampled metric tensors from (6). (b) The data represented in the tangent of a manifold constructed from local LDA metrics learned at random positions. **Right panels: Real data.** (c) Average error of linearly predicted body measurements (mm). (d) Running time (sec) of the geodesic computation as a function of dimensionality.

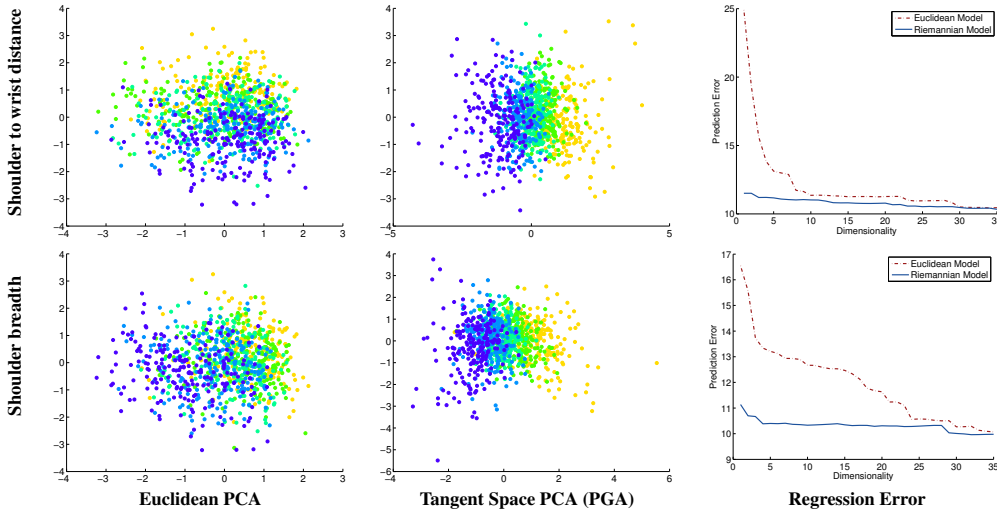

Figure 4: *Left:* body shape data in the first two principal components according to the Euclidean metric. Point color indicates cluster membership. *Center:* As on the left, but according to the Riemannian model. *Right:* regression error as a function of the dimensionality of the shape space; again the Euclidean metric and the Riemannian metric are compared.

compute the mean value on the manifold, map the data to the tangent space at the mean and perform linear regression in the tangent space.

As a first visualization we plot the data expressed in the leading two dimensions of PGA in Fig. 4; as can be seen the learned metrics provide principal geodesics, which are more strongly related with the measurements than the Euclidean model. In order to predict the measurements from the body shape, we perform linear regression, both directly in the shape space according to the Euclidean metric and in the tangent space of the manifold corresponding to the learned metrics (using the logarithmic map from (11)). We measure the prediction error using leave-one-out cross-validation. To further illustrate the power of the PGA model, we repeat this experiment for different dimensionalities of the data. The results are plotted in Fig. 4, showing that regression according to the learned metrics outperforms the Euclidean model.

To verify that the learned metrics improve accuracy, we average the prediction errors over all millimeter measurements. The result in Fig. 3c shows that much can be gained in lower dimensions by using the local metrics.

To provide visual insights into the behavior of the learned metrics, we uniformly sample body shape along the first principal geodesic (in the range ±7 times the standard deviation) according to the different metrics. The results are available as a movie in the supplementary material, but are also shown in Fig. 5. As can be seen, the learned metrics pick up intuitive relationships between body shape and the measurements, e.g. *shoulder to wrist distance* is related to overall body size, while *shoulder breadth* is related to body weight.

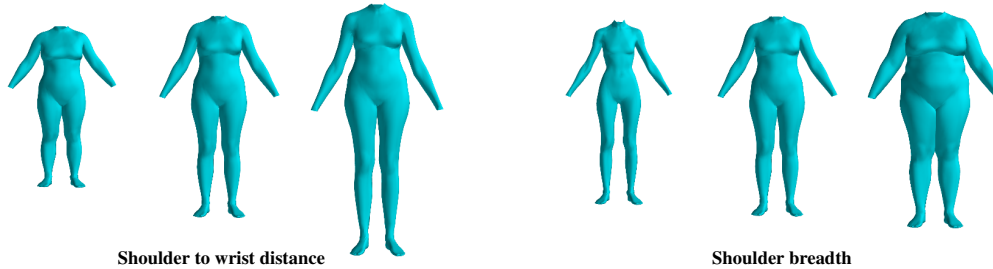

**Shoulder to wrist distance**  **Shoulder breadth**

Figure 5: Shapes corresponding to the mean (center) and $\pm 7$ times the standard deviations along the principal geodesics (left and right). Movies are available in the supplementary material.

## 4.2 Scalability

The human body data set is small enough (986 samples in 35 dimensions) that computing a geodesic only takes a few seconds. To show that the current unoptimized Matlab implementation can handle somewhat larger datasets, we briefly consider a dimensionality reduction task on the classic MNIST handwritten digit data set. We use the preprocessed data available with [3] where the original $28 \times 28$ gray scale images were deskewed and projected onto their leading $164$ Euclidean principal components (which captures 95% of the variance in the original data).

We learn one diagonal LMNN metric per class, which we associate with the mean of the class. From this we construct a Riemannian manifold from (6), compute the mean value on the manifold and compute geodesics between the mean and each data point; this is the computationally expensive part of performing PGA. Fig. 3d plots the average running time (sec) for the computation of geodesics as a function of the dimensionality of the training data. A geodesic can be computed in 100 dimensions in approximately 5 sec., whereas in 150 dimensions it takes about 30 sec.

In this experiment, we train a PGA model on 60,000 data points, and test a nearest neighbor classifier in the tangent space as we decrease the dimensionality of the model. Compared to a Euclidean model, this gives a modest improvement in classification accuracy of 2.3 percent, when averaged across different dimensionalities. Plots of the results can be found in the supplementary material.

## 5 Discussion

This work shows that multi-metric learning techniques are indeed applicable outside the realm of $k$NN classifiers. The idea of defining the metric tensor at any given point as the weighted average of a finite set of learned metrics is quite natural from a modeling point of view, which is also validated by the Riemannian structure of the resulting space. This opens both a theoretical and a practical toolbox for analyzing and developing algorithms that use local metric tensors. Specifically, we show how to use local metric tensors for both regression and dimensionality reduction tasks.

Others have attempted to solve non-classification problems using local metrics, but we feel that our approach is the first to have a solid theoretical backing. For example, Hastie and Tibshirani [9] use local LDA metrics for dimensionality reduction by averaging the local metrics and using the resulting metric as part of a Euclidean PCA, which essentially is a linear approach. Another approach was suggested by Hong et al. [23] who simply compute the principal components according to each metric separately, such that one low dimensional model is learned *per metric*.

The suggested approach is, however, not difficulty-free in its current implementation. Currently, we are using *off-the-shelf* numerical solvers for computing geodesics, which can be computationally demanding. While we managed to analyze medium-sized datasets, we believe that the run-time can be drastically improved by developing specialized numerical solvers.

In the experiments, we learned local metrics using techniques specialized for classification tasks as this is all the current literature provides. We expect improvements by learning the metrics specifically for regression and dimensionality reduction, but doing so is currently an open problem.

**Acknowledgments:** Søren Hauberg is supported in part by the Villum Foundation, and Oren Freifeld is supported in part by NIH-NINDS EUREKA (R01-NS066311).

## Footnotes

[1]Our software implementation for computing geodesics and performing manifold statistics is available at `http://ps.is.tue.mpg.de/project/Smooth_Metric_Learning`

# References

[1] Andrea Frome, Yoram Singer, and Jitendra Malik. Image retrieval and classification using local distance functions. In B. Schölkopf, J. Platt, and T. Hoffman, editors, *Advances in Neural Information Processing Systems 19 (NIPS)*, pages 417–424, Cambridge, MA, 2007. MIT Press.

[2] Andrea Frome, Fei Sha, Yoram Singer, and Jitendra Malik. Learning globally-consistent local distance functions for shape-based image retrieval and classification. In *International Conference on Computer Vision (ICCV)*, pages 1–8, 2007.

[3] Deva Ramanan and Simon Baker. Local distance functions: A taxonomy, new algorithms, and an evaluation. *IEEE Transactions on Pattern Analysis and Machine Intelligence*, 33(4):794–806, 2011.

[4] Shai Shalev-Shwartz, Yoram Singer, and Andrew Y. Ng. Online and batch learning of pseudo-metrics. In *Proceedings of the twenty-first international conference on Machine learning*, ICML '04, pages 94–101. ACM, 2004.

[5] Kilian Q. Weinberger and Lawrence K. Saul. Distance metric learning for large margin nearest neighbor classification. *The Journal of Machine Learning Research*, 10:207–244, 2009.

[6] Tomasz Malisiewicz and Alexei A. Efros. Recognition by association via learning per-exemplar distances. In *IEEE Conference on Computer Vision and Pattern Recognition (CVPR)*, pages 1–8, 2008.

[7] Yiming Ying and Peng Li. Distance metric learning with eigenvalue optimization. *The Journal of Machine Learning Research*, 13:1–26, 2012.

[8] Matthew Schultz and Thorsten Joachims. Learning a distance metric from relative comparisons. In *Advances in Neural Information Processing Systems 16 (NIPS)*, 2004.

[9] Trevor Hastie and Robert Tibshirani. Discriminant adaptive nearest neighbor classification. *IEEE Transactions on Pattern Analysis and Machine Intelligence*, 18(6):607–616, June 1996.

[10] Elzbieta Pekalska, Pavel Paclik, and Robert P. W. Duin. A generalized kernel approach to dissimilarity-based classification. *Journal of Machine Learning Research*, 2:175–211, 2002.

[11] Manfredo Perdigao do Carmo. *Riemannian Geometry*. Birkhäuser Boston, January 1992.

[12] Joshua B. Tenenbaum, Vin De Silva, and John C. Langford. A global geometric framework for nonlinear dimensionality reduction. *Science*, 290(5500):2319–2323, 2000.

[13] Jan R. Magnus and Heinz Neudecker. *Matrix Differential Calculus with Applications in Statistics and Econometrics*. John Wiley & Sons, 2007.

[14] Jacek Kierzenka and Lawrence F. Shampine. A BVP solver based on residual control and the Matlab PSE. *ACM Transactions on Mathematical Software*, 27(3):299–316, 2001.

[15] John R. Dormand and P. J. Prince. A family of embedded Runge-Kutta formulae. *Journal of Computational and Applied Mathematics*, 6:19–26, 1980.

[16] P. Thomas Fletcher, Conglin Lu, Stephen M. Pizer, and Sarang Joshi. Principal Geodesic Analysis for the study of Nonlinear Statistics of Shape. *IEEE Transactions on Medical Imaging*, 23(8):995–1005, 2004.

[17] Peter E. Jupp and John T. Kent. Fitting smooth paths to spherical data. *Applied Statistics*, 36(1):34–46, 1987.

[18] Xavier Pennec. Probabilities and statistics on Riemannian manifolds: Basic tools for geometric measurements. In *Proceedings of Nonlinear Signal and Image Processing*, pages 194–198, 1999.

[19] Stefan Sommer, François Lauze, Søren Hauberg, and Mads Nielsen. Manifold valued statistics, exact principal geodesic analysis and the effect of linear approximations. In *European Conference on Computer Vision (ECCV)*, pages 43–56, 2010.

[20] Kathleen M. Robinette, Hein Daanen, and Eric Paquet. The CAESAR project: a 3-D surface anthropometry survey. In *3-D Digital Imaging and Modeling*, pages 380–386, 1999.

[21] Dragomir Anguelov, Praveen Srinivasan, Daphne Koller, Sebastian Thrun, Jim Rodgers, and James Davis. Scape: shape completion and animation of people. *ACM Transactions on Graphics*, 24(3):408–416, 2005.

[22] Oren Freifeld and Michael J. Black. Lie bodies: A manifold representation of 3D human shape. In A. Fitzgibbon et al. (Eds.), editor, *European Conference on Computer Vision (ECCV)*, Part I, LNCS 7572, pages 1–14. Springer-Verlag, oct 2012.

[23] Yi Hong, Quannan Li, Jiayan Jiang, and Zhuowen Tu. Learning a mixture of sparse distance metrics for classification and dimensionality reduction. In *International Conference on Computer Vision (ICCV)*, pages 906–913, 2011.

